# Attribute-efficient learning
# of decision lists and linear threshold functions
# under unconcentrated distributions

**Philip M. Long**
Google
Mountain View, CA
plong@google.com

**Rocco A. Servedio**
Department of Computer Science
Columbia University
New York, NY
rocco@cs.columbia.edu

## Abstract

We consider the well-studied problem of learning decision lists using few examples when many irrelevant features are present. We show that smooth boosting algorithms such as MadaBoost can efficiently learn decision lists of length $k$ over $n$ boolean variables using $\text{poly}(k, \log n)$ many examples provided that the marginal distribution over the relevant variables is "not too concentrated" in an $L_2$-norm sense. Using a recent result of Håstad, we extend the analysis to obtain a similar (though quantitatively weaker) result for learning arbitrary linear threshold functions with $k$ nonzero coefficients. Experimental results indicate that the use of a *smooth* boosting algorithm, which plays a crucial role in our analysis, has an impact on the actual performance of the algorithm.

## 1   Introduction

A decision list is a Boolean function defined over $n$ Boolean inputs of the following form:

**if** $\ell_1$ **then** $b_1$ **else if** $\ell_2$ **then** $b_2$ ... **else if** $\ell_k$ **then** $b_k$ **else** $b_{k+1}$.

Here $\ell_1, ..., \ell_k$ are literals defined over the $n$ Boolean variables and $b_1, \ldots, b_{k+1}$ are Boolean values. Since the work of Rivest [24] decision lists have been widely studied in learning theory and machine learning.

A question that has received much attention is whether it is possible to *attribute-efficiently* learn decision lists, i.e. to learn decision lists of length $k$ over $n$ variables using only $\text{poly}(k, \log n)$ many examples. This question was first asked by Blum in 1990 [3] and has since been re-posed numerous times [4, 5, 6, 29]; as we now briefly describe, a range of partial results have been obtained along different lines.

Several authors [4, 29] have noted that Littlestone's Winnow algorithm [17] can learn decision lists of length $k$ using $2^{O(k)} \log n$ examples in time $2^{O(k)} n \log n$. Valiant [29] and Nevo and El-Yaniv [21] sharpened the analysis of Winnow in the special case where the decision list has only a bounded number of alternations in the sequence of output bits $b_1, \ldots, b_{k+1}$. It is well known that the "halving algorithm" (see [1, 2, 19]) can learn length-$k$ decision lists using only $O(k \log n)$ examples, but the running time of the algorithm is $n^k$. Klivans and Servedio [16] used polynomial threshold functions together with Winnow to obtain a tradeoff between running time and the number of examples required, by giving an algorithm that runs in time $n^{\tilde{O}(k^{1/3})}$ and uses $2^{\tilde{O}(k^{1/3})} \log n$ examples.

In this work we take a different approach by relaxing the requirement that the algorithm work under *any* distribution on examples or in the mistake-bound model. This relaxation in fact allows us to handle not just decision lists, but arbitrary *linear threshold functions* with $k$ nonzero coefficients. (Recall

that a linear threshold function $f : \{-1, 1\}^n \to \{-1, 1\}^n$ is a function $f(x) = \text{sgn}(\sum_{i=1}^n w_i x_i - \theta)$ where $w_i, \theta$ are real numbers and the sgn function outputs the $\pm 1$ numerical sign of its argument.)

**The approach and results.** We will analyze a *smooth* boosting algorithm (see Section 2) together with a weak learner that exhaustively considers all $2n$ possible literals $x_i, \neg x_i$ as weak hypotheses. The algorithm, which we call Algorithm $A$, is described in more detail in Section 6.

The algorithm's performance can be bounded in terms of the $L_2$-*norm* of the distribution over examples. Recall that the $L_2$-norm of a distribution $\mathcal{D}$ over a finite set $X$ is $\|\mathcal{D}\|_2 := (\sum_{x \in X} \mathcal{D}(x)^2)^{1/2}$. The $L_2$ norm can be used to evaluate the "spread" of a probability distribution: if the probability is concentrated on a constant number of elements of the domain then the $L_2$ norm is constant, whereas if the probability mass is spread uniformly over a domain of size $N$ then the $L_2$ norm is $1/\sqrt{N}$.

Our main results are as follows. Let $\mathcal{D}$ be a distribution over $\{-1, 1\}^n$. Suppose the target function $f$ has $k$ relevant variables. Let $\mathcal{D}^{rel}$ denote the marginal distribution over $\{-1, 1\}^k$ induced by the relevant variables to $f$ (i.e. if the relevant variables are $x_{i_1}, \ldots, x_{i_k}$, then the value that $\mathcal{D}^{rel}$ puts on an input $(z_1, \ldots, z_k)$ is $\Pr_{x \in \mathcal{D}}[x_{i_1} \ldots x_{i_k} = z_1 \ldots z_k]$. Let $\mathcal{U}_k$ be the uniform distribution over $\{-1, 1\}^k$ and suppose that $\|\mathcal{D}^{rel}\|_2 / \|\mathcal{U}_k\|_2 = \tau$. (Note that for any $\mathcal{D}$ we have $\tau \geq 1$, since $\mathcal{U}_k$ has minimal $L_2$-norm among all distributions over $\{-1, 1\}^k$.) Then we have:

**Theorem 1** *Suppose the target function is an arbitrary decision list in the setting described above. Then given poly($\log n, \frac{1}{\epsilon}, \tau, \log \frac{1}{\delta}$) examples, Algorithm $A$ runs in poly($n, \tau, \frac{1}{\epsilon}, \log \frac{1}{\delta}$) time and with probability $1 - \delta$ constructs a hypothesis $h$ that is $\epsilon$-accurate with respect to $\mathcal{D}$.*

**Theorem 2** *Suppose the target function is an arbitrary linear threshold function in the setting described above. Then given poly($k, \log n, 2^{\tilde{O}((\tau/\epsilon)^2)}, \log \frac{1}{\delta}$) examples, Algorithm $A$ runs in poly($n$, $2^{\tilde{O}((\tau/\epsilon)^2)}, \log \frac{1}{\delta}$) time and with probability $1 - \delta$ constructs a hypothesis $h$ that is $\epsilon$-accurate with respect to $\mathcal{D}$.*

**Relation to Previous Work.** Jackson and Craven [14] considered a similar approach of using Boolean literals as weak hypotheses for a boosting algorithm (in their case, AdaBoost). Jackson and Craven proved that for any distribution over examples, the resulting algorithm requires poly($K, \log n$) examples to learn any *weight-$K$* linear threshold function, i.e. any function of the form $\text{sgn}(\sum_{i=1}^n w_i x_i - \theta)$ over Boolean variables where all weights $w_i$ are integers and $\sum_{i=1}^n |w_i| \leq K$ (this clearly implies that there are at most $K$ relevant variables). It is well known [12, 18] that general decision lists of length $k$ can only be expressed by linear threshold functions of weight $2^{\Omega(k)}$, and thus the result of [14] does not give an attribute efficient learning algorithm for decision lists.

More recently Servedio [27] considered essentially the same algorithm we analyze in this work by specifically studying smooth boosting algorithms with the "best-single-variable" weak learner. He considered a general linear threshold learning problem (with no assumption that there are few relevant variables) and showed that if the distribution satisfies a margin condition then the algorithm has some level of resilience to malicious noise. The analysis of this paper is different from that of [27]; to the best of our knowledge ours is the first analysis in which the smoothness property of boosting is exploited for attribute efficient learning.

## 2  Boosting and Smooth Boosting

Fix a target function $f : \{-1, 1\}^n \to \{-1, 1\}$ and a distribution $\mathcal{D}$ over $\{-1, 1\}^n$. A hypothesis function $h : \{-1, 1\}^n \to \{-1, 1\}$ is a $\gamma$-*weak hypothesis for $f$ with respect to $\mathcal{D}$* if $\mathbf{E}_{\mathcal{D}}[fh] \geq \gamma$. We sometimes refer to $\mathbf{E}_{\mathcal{D}}[fh]$ as the *advantage* of $h$ with respect to $f$.

We remind the reader that a *boosting algorithm* is an algorithm which operates in a sequence of stages and at each stage $t$ maintains a distribution $\mathcal{D}_t$ over $\{-1, 1\}^n$. At stage $t$ the boosting algorithm is given a weak hypothesis $h_t$ for $f$ with respect to $\mathcal{D}$; the boosting algorithm then uses this to construct the next distribution $\mathcal{D}_{t+1}$ over $\{-1, 1\}^n$. After $T$ such stages the boosting algorithm constructs a final hypothesis $h$ based on the weak hypotheses $h_1, \ldots, h_T$ that is guaranteed to have high accuracy with respect to the initial distribution $\mathcal{D}$. See [25] for more details.

Let $\mathcal{D}_1, \mathcal{D}_2$ be two distributions. For $\kappa \geq 1$ we say that $\mathcal{D}_1$ is $\kappa$-*smooth with respect to* $\mathcal{D}_2$ if
$$\text{for all } x \in \{-1,1\}^n, \quad \mathcal{D}_1(x)/\mathcal{D}_2(x) \leq \kappa.$$

Following [15], we say that a boosting algorithm $\mathcal{B}$ is $\kappa(\epsilon, \gamma)$-*smooth* if for any initial distribution $\mathcal{D}$ and any distribution $\mathcal{D}_t$ that is generated starting from $\mathcal{D}$ when $\mathcal{B}$ is used to boost to $\epsilon$-accuracy with $\gamma$-weak hypotheses at each stage, $\mathcal{D}_t$ is $\kappa(\epsilon, \gamma)$-smooth w.r.t. $\mathcal{D}$. It is known that there are algorithms that are $\kappa$-smooth for $\kappa = \Theta(\frac{1}{\epsilon})$ with no dependence on $\gamma$, see e.g. [8]. For the rest of the paper $\mathcal{B}$ will denote such a smooth boosting algorithm.

It is easy to see that every distribution $\mathcal{D}$ which is $\frac{1}{\epsilon}$-smooth w.r.t. the uniform distribution $\mathcal{U}$ satisfies $\|\mathcal{D}\|_2/\|\mathcal{U}\|_2 \leq \sqrt{1/\epsilon}$. On the other hand, there are distributions $\mathcal{D}$ that are highly non-smooth relative to $\mathcal{U}$ but which still have $\|\mathcal{D}\|_2/\|\mathcal{U}\|_2$ small. For instance, the distribution $\mathcal{D}$ over $\{-1,1\}^k$ which puts weight $\frac{1}{2^{k/2}}$ on a single point and distributes the remaining weight uniformly on the other $2^k - 1$ points is only $2^{k/2}$-smooth (i.e. very non-smooth) but satisfies $\|\mathcal{D}\|_2/\|\mathcal{U}_k\|_2 = \Theta(1)$. Thus the $L_2$-norm condition we consider in this paper is a weaker condition than smoothness with respect to the uniform distribution.

## 3  Total variation distance and $L_2$-norm of distributions

The *total variation distance* between two probability distributions $\mathcal{D}_1, \mathcal{D}_2$ over a finite set $X$ is $d_{TV} := \max_{S \subseteq X} \mathcal{D}_1(S) - \mathcal{D}_2(S) = \frac{1}{2}\sum_{x \in X}|\mathcal{D}_1(x) - \mathcal{D}_2(x)|$. It is easy to see that the total variation distance between any two distributions is at most 1, and equals 1 if and only if the supports of the distributions are disjoint. The following is immediate:

**Lemma 1** *For any two distributions $\mathcal{D}_1$ and $\mathcal{D}_2$ over a finite domain $X$, we have $d_{TV}(\mathcal{D}_1, \mathcal{D}_2) = 1 - \sum_{x \in X} \min\{\mathcal{D}_1(x), \mathcal{D}_2(x)\}$.*

We can bound the total variation distance between a distribution $\mathcal{D}$ and the uniform distribution in terms of the ratio $\|\mathcal{D}\|_2/\|\mathcal{U}\|_2$ of the $L_2$-norms as follows:

**Lemma 2** *For any distribution $\mathcal{D}$ over a finite domain $X$, if $\mathcal{U}$ is the uniform distribution over $X$, we have $d_{TV}(\mathcal{D}, \mathcal{U}) \leq 1 - \frac{\|\mathcal{U}\|_2^2}{4\|\mathcal{D}\|_2^2}$.*

**Proof:** Let $M = \frac{\|\mathcal{D}\|_2}{\|\mathcal{U}\|_2}$. Since $\|\mathcal{D}\|_2^2 = \mathbf{E}_{x \sim D}[D(x)]$, we have $\mathbf{E}_{x \sim D}[D(x)] = M^2\|\mathcal{U}\|_2^2 = \frac{M^2}{|X|}$. By Markov's inequality,

$$\Pr_{x \sim D}[\mathcal{D}(x) \geq 2M^2 \mathcal{U}(x)] = \Pr_{x \sim D}[\mathcal{D}(x) \geq \frac{2M^2}{|X|}] \leq 1/2. \tag{1}$$

By Lemma 1, we have

$$
\begin{aligned}
1 - d_{TV}(\mathcal{D}, \mathcal{U}) &= \sum_x \min\{\mathcal{D}(x), \mathcal{U}(x)\} \geq \sum_{x:\mathcal{D}(x) \leq 2M^2\mathcal{U}(x)} \min\{\mathcal{D}(x), \mathcal{U}(x)\} \\
&\geq \sum_{x:\mathcal{D}(x) \leq 2M^2\mathcal{U}(x)} \frac{\mathcal{D}(x)}{2M^2} \geq \frac{1}{4M^2},
\end{aligned}
$$

where the second inequality uses the fact that $M \geq 1$ (so $\mathcal{D}(x)/2M^2 < \mathcal{D}(x)$) and the third inequality uses (1). Using the definition of $M$ and solving for $d_{TV}(\mathcal{D}, \mathcal{U})$ completes the proof. ∎

## 4  Weak hypotheses for decision lists

Let $f$ be any decision list that depends on $k$ variables:

$$\textbf{if } \ell_1 \textbf{ then output } b_1 \textbf{ else } \cdots \textbf{ else if } \ell_k \textbf{ then output } b_k \textbf{ else output } b_{k+1} \tag{2}$$

where each $\ell_i$ is either "$(x_i = 1)$" or "$(x_i = -1)$."

The following folklore lemma can be proved by an easy induction (see e.g. [12, 26] for proofs of essentially equivalent claims):

**Lemma 3** *The decision list $f$ can be represented by a linear threshold function of the form $f(x) = \text{sgn}(c_1 x_1 + \cdots + c_k x_k - \theta)$ where each $c_i = \pm 2^{k-i}$ and $\theta$ is an even integer in the range $[-2^k, 2^k]$.*

It is easy to see that for any fixed $c_1, \ldots, c_k$ as in the lemma, as $x = (x_1, \ldots, x_k)$ varies over $\{-1, 1\}^k$ the linear form $c_1 x_1 + \cdots + c_k x_k$ will assume each odd integer value in the range $[-2^k, 2^k]$ exactly once. Now we can prove:

**Lemma 4** *Let $f$ be any decision list of length $k$ over the $n$ Boolean variables $x_1, \ldots, x_n$. Let $\mathcal{D}$ be any distribution over $\{-1, 1\}^n$, and let $\mathcal{D}^{rel}$ denote the marginal distribution over $\{-1, 1\}^k$ induced by the $k$ relevant variables of $f$. Suppose that $d_{TV}(\mathcal{D}^{rel}, \mathcal{U}_k) \leq 1 - \eta$. Then there is some weak hypothesis $h \in \{x_1, -x_1, \ldots, x_n, -x_n, 1, -1\}$ which satisfies $\mathbf{E}_{\mathcal{D}^{rel}}[fh] \geq \frac{\eta^2}{16}$.*

**Proof:** We first observe that by Lemma 3 and the well-known "discriminator lemma" of [23, 11], under *any* distribution $\mathcal{D}$ some weak hypothesis $h$ from $\{x_1, -x_1, \ldots, x_n, -x_n, 1, -1\}$ must have $\mathbf{E}_{\mathcal{D}}[fh] \geq \frac{1}{2^k}$. This immediately establishes the lemma for all $\eta \leq \frac{4}{2^{k/2}}$, and thus we may suppose w.l.o.g. that $\eta > \frac{4}{2^{k/2}}$.

We may assume w.l.o.g. that $f$ is the decision list (2), that is, that the first literal concerns $x_1$, the second concerns $x_2$, and so on. Let $L(x)$ denote the linear form $c_1 x_1 + \cdots + c_k x_k - \theta$ from Lemma 3, so $f(x) = \text{sgn}(L(x))$. If $x$ is drawn uniformly from $\{-1, 1\}^k$, then $L(x)$ is distributed uniformly over the $2^k$ odd integers in the interval $[-2^k - \theta, 2^k - \theta]$, as $c_1 x_1$ is uniform over $\pm 2^k$, $c_2 x_2$ over $\pm 2^{k-1}$, and so on.

Let $S$ denote the set of those $x \in \{-1, 1\}^k$ that satisfy $|L(x)| \leq \frac{\eta}{4} 2^k$. Note that there are at most $\frac{\eta}{4} 2^k + 1$ elements in $S$, corresponding to $L(x) = \pm 1, \pm 3, \ldots, \pm(2j - 1)$, where $j$ is the greatest integer such that $2j - 1 \leq \frac{\eta}{4} 2^k$. Since $\eta > \frac{4}{2^{k/2}}$, certainly $|S| \leq 1 + \frac{\eta}{4} 2^k \leq \frac{\eta}{2} 2^k$. We thus have $\text{Pr}_{\mathcal{U}_k}[|L(x)| > \frac{\eta}{4} 2^k] \geq 1 - \eta/2$. It follows that $\text{Pr}_{\mathcal{D}^{rel}}[|L(x)| > \frac{\eta}{4} 2^k] \geq \frac{\eta}{2}$ (for otherwise we would have $d_{TV}(\mathcal{D}^{rel}, \mathcal{U}_k) > 1 - \eta$), and consequently we have $\mathbf{E}_{\mathcal{D}^{rel}}[|L(x)|] \geq \frac{\eta^2}{8} 2^k$.

Now we follow the simple argument used to prove the "discriminator lemma" [23, 11]. We have

$$\mathbf{E}_{\mathcal{D}^{rel}}[|L(x)|] = \mathbf{E}_{\mathcal{D}^{rel}}[f(x)L(x)] = c_1 \mathbf{E}[f(x)x_1] + \cdots + c_k \mathbf{E}[f(x)x_k] - \theta \mathbf{E}[f(x)] \geq \frac{\eta^2}{8} 2^k. \quad (3)$$

Recalling that each $|c_i| = 2^{k-i}$, it follows that some $h \in \{x_1, -x_1, \ldots, x_n, -x_n, 1, -1\}$ must satisfy $\mathbf{E}_{\mathcal{D}^{rel}}[fh] \geq (\frac{\eta^2}{8} 2^k)/(2^{k-1} + \cdots + 2^0 + |\theta|)$. Since $|\theta| \leq 2^k$ this is at least $\frac{\eta^2}{16}$, and the proof is complete. ∎

## 5  Weak hypotheses for linear threshold functions

Now we consider the more general setting of arbitrary linear threshold functions. Though there are additional technical complications the basic idea is as in the previous section.

We will use the following fact due to Håstad:

**Fact 3 (Håstad)** *(see [28], Theorem 9) Let $f : \{-1, 1\}^k \to \{-1, 1\}$ be any linear threshold function that depends on all $k$ variables $x_1, \ldots, x_k$. There is a representation $\text{sgn}(\sum_{i=1}^k w_i x_i - \theta)$ for $f$ which is such that (assuming the weights $w_1, \ldots, w_k$ are ordered by decreasing magnitude $1 = |w_1| \geq |w_2| \geq \cdots \geq |w_k| > 0$) we have $|w_i| \geq \frac{1}{i!(k+1)}$ for all $i = 2, \ldots, k$.*

The main result of this section is the following lemma. The proof uses ideas from the proof of Theorem 2 in [28].

**Lemma 5** *Let $f : \{-1, 1\}^n \to \{-1, 1\}$ be any linear threshold function that depends on $k$ variables. Let $\mathcal{D}$ be any distribution over $\{-1, 1\}^n$, and let $\mathcal{D}^{rel}$ denote the marginal distribution over $\{-1, 1\}^k$ induced by the $k$ relevant variables of $f$. Suppose that $d_{TV}(\mathcal{D}^{rel}, \mathcal{U}_k) \leq 1 - \eta$. Then there is some weak hypothesis $h \in \{x_1, -x_1, \ldots, x_n, -x_n, 1, -1\}$ which satisfies $\mathbf{E}_{\mathcal{D}^{rel}}[fh] \geq 1/(k^2 2^{\tilde{O}(1/\eta^2)})$.*

**Proof sketch:** We may assume that $f(x) = \text{sgn}(L(x))$ where $L(x) = w_1 x_1 + \cdots + w_k x_k - \theta$ with $w_1, \ldots, w_k$ as described in Fact 3.

Let $\ell := \tilde{O}(1/\eta^2) = O((1/\eta^2)\text{poly}(\log(1/\eta)))$. (We will specify $\ell$ in more detail later.)

Suppose first that $\ell \geq k$. By a well-known result of Muroga *et al.* [20], every linear threshold function $f$ that depends on $k$ variables can be represented using integer weights each of magnitude $2^{O(k \log k)}$. Now the discriminator lemma [11] implies that for *any* distribution $\mathcal{P}$, for some $h \in \{x_1, -x_1, \ldots, x_n, -x_n, 1, -1\}$ we have $\mathbf{E}_{\mathcal{P}}[fh] \geq 1/2^{O(k \log k)}$. If $\ell \geq k$ and $\ell = O((1/\eta^2)\text{poly}(\log(1/\eta)))$, we have $k \log k = \tilde{O}(1/\eta^2)$. Thus, in this case, $\mathbf{E}_{\mathcal{P}}[fh] \geq 1/2^{\tilde{O}(1/\eta^2)}$, so the lemma holds if $\ell \geq k$.

Thus we henceforth assume that $\ell < k$. It remains only to show that

$$\mathbf{E}_{\mathcal{D}^{rel}}[|L(x)|] \geq 1/(k2^{\tilde{O}(1/\eta^2)}); \tag{4}$$

once we have this, following (3) we get

$$\mathbf{E}_{\mathcal{D}^{rel}}[|L(x)|] = \mathbf{E}_{\mathcal{D}^{rel}}[fL] = w_1\mathbf{E}[f(x)x_1] + \cdots + w_k\mathbf{E}[f(x)x_k] - \theta\mathbf{E}[f(x)] \geq 1/(k2^{\tilde{O}(1/\eta^2)}),$$

and now since each $|w_i| \leq 1$ (and w.l.o.g. $|\theta| \leq k$) this implies that some $h$ satisfies $\mathbf{E}_{\mathcal{D}^{rel}}[fh] \geq 1/(k^2 2^{\tilde{O}(1/\eta^2)})$ as desired.

Similar to [28] we consider two cases (which are slightly different from the cases in [28]).

**Case I:** For all $1 \leq i \leq \ell$ we have $w_i^2/(\sum_{j=i}^k w_j^2) > \eta^2/576$.

Let $\alpha := \sqrt{2\left(\sum_{j=\ell+1}^k w_j^2\right)\ln(8/\eta)}$. Recall the following version of Hoeffding's bound: for any $0 \neq w \in \mathbf{R}^k$ and any $\gamma > 0$, we have $\text{Pr}_{x \in \{-1,1\}^k}[|w \cdot x| \geq \gamma\|w\|] \leq 2e^{-\gamma^2/2}$ (where we write $\|w\|$ to denote $\sqrt{\sum_{i=1}^k w_i^2}$). This bound directly gives us that

$$\Pr_{x \in \mathcal{U}_k}[|w_{\ell+1}x_{\ell+1} + \cdots + w_k x_k| \geq \alpha] \leq 2e^{-2\ln(8/\eta)/2} = \frac{\eta}{4}. \tag{5}$$

Moreover, the argument in [28] that establishes equation (4) of [28] also yields

$$\Pr_{x \in \mathcal{U}_k}[|w_1 x_1 + \cdots + w_\ell x_\ell - \theta| \leq 2\alpha] \leq \frac{\eta}{4} \tag{6}$$

in our current setting. (The only change that needs to be made to the argument of [28] is adjusting various constant factors in the definition of $\ell$.) Equations (5) and (6) together yield $\text{Pr}_{x \in \mathcal{U}_k}[|w_1 x_1 + \cdots + w_k x_k - \theta| \geq \alpha] \geq 1 - \frac{\eta}{2}$. Now as before, taken together with the $d_{TV}$ bound this yields $\text{Pr}_{\mathcal{D}^{rel}}[|L(x)| \geq \alpha] \geq \frac{\eta}{2}$ and hence we have $\mathbf{E}_{\mathcal{D}^{rel}}[|L(x)|] \geq \eta\alpha/2$. Since $\alpha > w_{\ell+1}$ and $w_{\ell+1} \geq 1/((k+1)(\ell+1)!)$ by Fact 3, we have established (4) in Case I.

**Case II:** For some value $J \leq \ell$ we have $w_J^2/(\sum_{i=J}^k w_i^2) \leq \eta^2/576$. Let us fix any setting $z \in \{-1,1\}^{J-1}$ of the variables $x_1, \ldots, x_{J-1}$. By an inequality due to Petrov [22] (see [28], Theorem 4) we have

$$\Pr_{x_J,\ldots,x_k \in \mathcal{U}_{k-J+1}}[|w_1 z_1 + \cdots + w_{J-1}z_{J-1} + w_J x_J + \cdots + w_k x_k - \theta| \leq w_J] \leq \frac{6w_J}{\sqrt{\sum_{i=J}^k w_i^2}} \leq \frac{6\eta}{24} = \frac{\eta}{4}.$$

Thus for each $z \in \{-1,1\}^{J-1}$ we have $\text{Pr}_{x \in \mathcal{U}_k}[|L(x)| \leq w_J \mid x_1 \ldots x_{J-1} = z_1 \ldots z_{J-1}] \leq \frac{\eta}{4}$. This immediately yields $\text{Pr}_{x \in \mathcal{U}_k}[|L(x)| > w_J] \geq 1 - \frac{\eta}{4}$, which in turn gives $\text{Pr}_{x \in \mathcal{D}^{rel}}[|L(x)| > w_J] \geq \frac{3\eta}{4}$ and hence $\mathbf{E}_{\mathcal{D}^{rel}}[|L(x)|] \geq \frac{3\eta w_J}{4}$ by our usual arguments. Now (4) follows using Fact 3 and $J \leq \ell$. ∎

# 6 Putting it all together

Algorithm $A$ works by running a $\Theta(\frac{1}{\epsilon})$-smooth boosting-by-filtering algorithm; for concreteness we use the MadaBoost algorithm of Domingo and Watanabe [8]. At the $t$-th stage of boosting,

when MadaBoost simulates the distribution $\mathcal{D}_t$, the weak learning algorithm works as follows: $O(\frac{\log n + \log(1/\delta')}{\gamma^2})$ many examples are drawn from the simulated distribution $\mathcal{D}_t$, and these examples are used to obtain an empirical estimate of $\mathbf{E}_{\mathcal{D}_t}[fh]$ for each $h \in \{x_1, -x_1, \ldots, x_n, -x_n, -1, 1\}$. (Here $\gamma$ is an upper bound on the advantage $\mathbf{E}_{\mathcal{D}_t}[fh]$ of the weak hypotheses used at each stage; we discuss this more below.) The weak hypothesis used at this stage is the one with the highest observed empirical estimate. The algorithm is run for $T = O(\frac{1}{\epsilon \gamma^2})$ stages of boosting.

Consider any fixed stage $t$ of the algorithm's execution. As shown in [8], at most $O(\frac{1}{\epsilon})$ draws from the original distribution $\mathcal{D}$ are required for MadaBoost to simulate a draw from the distribution $\mathcal{D}_t$. (This is a direct consequence of the fact that MadaBoost is $O(\frac{1}{\epsilon})$-smooth; the distribution $\mathcal{D}_t$ is simulated using rejection sampling from $\mathcal{D}$.) Standard tail bounds show that if the best hypothesis $h$ has $\mathbf{E}[fh] \geq \gamma$ then with probability $1 - \delta'$ the hypothesis selected will have $\mathbf{E}[fh] \geq \gamma/2$. In [8] it is shown that if MadaBoost always has an $\Omega(\gamma)$-accurate weak hypothesis at each stage, then after at most $T = O(\frac{1}{\epsilon \gamma^2})$ stages the algorithm will construct a hypothesis which has error at most $\epsilon$. Thus it suffices to take $\delta' = O(\delta \epsilon^2 \gamma)$. The overall number of examples used by Algorithm $A$ is $O(\frac{\log n + \log(1/\delta')}{\epsilon^2 \gamma^4})$.

Thus to establish Theorems 1 and 2, it remains only to show that for any initial distribution $\mathcal{D}$ with $\|\mathcal{D}^{rel}\|_2/\|\mathcal{U}_k\|_2 = \tau$, the distributions $\mathcal{D}_t$ that arise in the course of boosting are always such that the best weak hypothesis $h \in \{x_1, -x_1, \ldots, x_n, -x_n, -1, 1\}$ has sufficiently large advantage.

Suppose $f$ is a target function that depends on some set of $k$ (out of $n$) variables. Consider what happens if we run a $\frac{1}{\epsilon}$-smooth boosting algorithm, where the initial distribution $\mathcal{D}$ satisfies $\|\mathcal{D}^{rel}\|/\|\mathcal{U}_k\| = \tau$. At each stage we will have $\mathcal{D}_t^{rel}(x) \leq \frac{1}{\epsilon} \cdot \mathcal{D}^{rel}(x)$ for all $x \in \{-1, 1\}^k$, and consequently we will have

$$\|\mathcal{D}_t^{rel}\|_2^2 = \sum_{x \in \{-1,1\}^k} \mathcal{D}_t^{rel}(x)^2 \leq \frac{1}{\epsilon^2} \sum_{x \in \{-1,1\}^k} \mathcal{D}^{rel}(x)^2 \leq \frac{\tau^2}{\epsilon^2} \sum_{x \in \{-1,1\}^k} \mathcal{U}_k(x)^2.$$

Thus, by Lemma 2 each distribution $\mathcal{D}_t$ will satisfy $d_{TV}(\mathcal{D}_t^{rel}, \mathcal{U}_k) \leq 1 - \epsilon^2/(4\tau^2)$. Now Lemmas 4 and 5 imply that in both cases (decision lists and LTFs) the best weak hypothesis $h$ does indeed have the required advantage.

## 7 Experiments

The smoothness property enabled the analysis of this paper. Is smoothness really helpful for learning decision lists with respect to diffuse distributions? Is it critical?

This section is aimed at addressing these questions experimentally. We compared the accuracy of the classifiers output by a number of smooth boosters from the literature with AdaBoost (which is known to not be a smooth booster in general, see e.g. Section 4.2 of [7]) on synthetic data in which the examples were distributed uniformly, and the class designations were determined by applying a randomly generated decision list. The number of relevant variables was fixed at 10. The decision list was determined by picking $\ell_1, ..., \ell_{10}$ and $b_1, ..., b_{11}$ from (2) independently uniformly at random from among the possibilities.

We evaluated the following algorithms: (a) AdaBoost [9], (b) MadaBoost [8], (c) SmoothBoost [27], and (d) a smooth booster proposed by Gavinsky [10]. Due to space constraints, we cannot describe each of these in detail.[1]

Each booster was used to reweight the training data, and in each round, the literal which minimized the weighted training error was chosen. Some of the algorithms choose the number of rounds of

| $m$ | $n$ | Ada | Mada | Gavinsky | SB(0.05) | SB(0.1) | SB(0.2) | SB(0.4) |
|---|---|---|---|---|---|---|---|---|
| 100 | 100 | 0.086 | 0.077 | 0.088 | 0.071 | 0.067 | 0.077 | 0.089 |
| 200 | 100 | 0.052 | 0.045 | 0.050 | 0.067 | 0.047 | 0.047 | 0.051 |
| 500 | 100 | 0.022 | 0.018 | 0.024 | 0.056 | 0.031 | 0.025 | 0.031 |
| 1000 | 100 | 0.016 | 0.014 | 0.024 | 0.063 | 0.036 | 0.028 | 0.033 |
| 100 | 1000 | 0.123 | 0.119 | 0.116 | 0.093 | 0.101 | 0.117 | 0.128 |
| 200 | 1000 | 0.079 | 0.072 | 0.083 | 0.071 | 0.064 | 0.072 | 0.081 |
| 500 | 1000 | 0.045 | 0.039 | 0.045 | 0.050 | 0.040 | 0.040 | 0.044 |
| 1000 | 1000 | 0.033 | 0.026 | 0.035 | 0.048 | 0.038 | 0.032 | 0.036 |

Table 1: Average test set error rate

| $m$ | $n$ | Ada | Mada | Gavinsky | SB(0.05) | SB(0.1) | SB(0.2) | SB(0.4) |
|---|---|---|---|---|---|---|---|---|
| 100 | 100 | 13.6 | 8.8 | 11.7 | 3.9 | 6.0 | 7.5 | 9.1 |
| 200 | 100 | 19.8 | 13.1 | 12.5 | 4.1 | 6.9 | 9.4 | 9.9 |
| 500 | 100 | 32.2 | 20.7 | 15.2 | 5.0 | 9.1 | 11.5 | 12.2 |
| 1000 | 100 | 37.2 | 19.2 | 15.3 | 7.1 | 10.7 | 12.1 | 13.0 |
| 100 | 1000 | 13.3 | 7.7 | 26.8 | 3.7 | 5.3 | 6.1 | 7.4 |
| 200 | 1000 | 19.8 | 11.5 | 19.4 | 4.4 | 7.4 | 9.5 | 11.7 |
| 500 | 1000 | 28.1 | 16.7 | 16.2 | 4.9 | 8.6 | 10.9 | 11.5 |
| 1000 | 1000 | 36.7 | 20.1 | 14.7 | 7.2 | 11.0 | 12.1 | 13.3 |

Table 2: Average smoothness

boosting as a function of the desired accuracy; instead, we ran all algorithms for 100 rounds. All boosters reweighted the data by normalizing some function that assigns weight to examples based on how well previously chosen based classifiers are doing at classifying them correctly. The booster proposed by Gavinsky might set all of these weights to zero: in such cases, it was terminated.

For each choice of the number of examples $m$ and the number of features $n$, we repeated the following steps: (a) generate a random target, (b) generate $m$ random examples, (c) split them into a training set with $2/3$ of the examples and a test set with the remaining $1/3$, (d) apply all the algorithms on the training set, and (e) apply all the resulting classifiers on the test set. We repeated the steps enough times so that the total size of the test sets was at least $10000$; that is, we repeated them $\lceil 30000/m \rceil$ times. The average test-set error is reported.

SmoothBoost [27] has two parameters, $\gamma$ and $\theta$. In his analysis, $\theta = \gamma/(2+\gamma)$, so we used the same setting. We tried his algorithm with $\gamma$ set to each of $0.05$, $0.1$, $0.2$ and $0.4$.

The test set error rates are tabulated in Table 1. MadaBoost always improved on the accuracy of AdaBoost. The results are consistent with the possibility that AdaBoost learns decision lists attribute-efficiently with respect to the uniform distribution; this motivates theoretical study of whether this is true. One possible route is to prove that, for sources like this, AdaBoost is, with high probability, a smooth boosting algorithm. The average smoothnesses are given in Table 2.

SmoothBoost [27] was seen to be fairly robust to the choice of $\gamma$; with a good choice it sometimes performed the best. This motivates research into adaptive boosters along the lines of SmoothBoost.

## Footnotes

[1]Very roughly speaking, AdaBoost reweights the data to assign more weight to examples that previously chosen base classifiers have often classified incorrectly; it then outputs a weighted vote over the outputs of the base classifiers, where each voting weight is determined as a function of how well its base classifier performed. MadaBoost modifies AdaBoost to place a cap on the weight, prior to normalization. SmoothBoost [27] caps the weight more aggressively as learning progresses, but also reweights the data and weighs the base classifiers in a manner that does not depend on how well they performed. The form of the manner in which Gavinsky's booster updates weights is significantly different from AdaBoost, and reminiscent of [13, 15].

## References

[1] D. Angluin. Queries and concept learning. *Machine Learning*, 2:319–342, 1988.

[2] J. Barzdin and R. Freivald. On the prediction of general recursive functions. *Soviet Mathematics Doklady*, 13:1224–1228, 1972.

[3] A. Blum. Learning Boolean functions in an infinite attribute space. In *Proceedings of the Twenty-Second Annual Symposium on Theory of Computing*, pages 64–72, 1990.

[4] A. Blum. On-line algorithms in machine learning. available at http://www.cs.cmu.edu/~avrim/Papers/pubs.html, 1996.

[5] A. Blum, L. Hellerstein, and N. Littlestone. Learning in the presence of finitely or infinitely many irrelevant attributes. *Journal of Computer and System Sciences*, 50:32–40, 1995.

[6] A. Blum and P. Langley. Selection of relevant features and examples in machine learning. *Artificial Intelligence*, 97(1-2):245–271, 1997.

[7] N. Bshouty and D. Gavinsky. On boosting with optimal poly-bounded distributions. *Journal of Machine Learning Research*, 3:483–506, 2002.

[8] C. Domingo and O. Watanabe. Madaboost: a modified version of adaboost. In *Proceedings of the Thirteenth Annual Conference on Computational Learning Theory*, pages 180–189, 2000.

[9] Y. Freund and R. Schapire. A decision-theoretic generalization of on-line learning and an application to boosting. *Journal of Computer and System Sciences*, 55(1):119–139, 1997.

[10] Dmitry Gavinsky. Optimally-smooth adaptive boosting and application to agnostic learning. *Journal of Machine Learning Research*, 4:101–117, 2003.

[11] A. Hajnal, W. Maass, P. Pudlak, M. Szegedy, and G. Turan. Threshold circuits of bounded depth. *Journal of Computer and System Sciences*, 46:129–154, 1993.

[12] S. Hampson and D. Volper. Linear function neurons: structure and training. *Biological Cybernetics*, 53:203–217, 1986.

[13] R. Impagliazzo. Hard-core distributions for somewhat hard problems. In *Proceedings of the Thirty-Sixth Annual Symposium on Foundations of Computer Science*, pages 538–545, 1995.

[14] J. Jackson and M. Craven. Learning sparse perceptrons. In *NIPS 8*, pages 654–660, 1996.

[15] A. Klivans and R. Servedio. Boosting and hard-core sets. *Machine Learning*, 53(3):217–238, 2003. Preliminary version in *Proc. FOCS'99*.

[16] A. Klivans and R. Servedio. Toward attribute efficient learning of decision lists and parities. In *Proceedings of the 17th Annual Conference on Learning Theory,*, pages 224–238, 2004.

[17] N. Littlestone. Learning quickly when irrelevant attributes abound: a new linear-threshold algorithm. *Machine Learning*, 2:285–318, 1988.

[18] M. Minsky and S. Papert. *Perceptrons: an introduction to computational geometry*. MIT Press, Cambridge, MA, 1968.

[19] T. Mitchell. Generalization as search. *Artificial Intelligence*, 18:203–226, 1982.

[20] S. Muroga, I. Toda, and S. Takasu. Theory of majority switching elements. *J. Franklin Institute*, 271:376–418, 1961.

[21] Z. Nevo and R. El-Yaniv. On online learning of decision lists. *Journal of Machine Learning Research*, 3:271–301, 2002.

[22] V. V. Petrov. *Limit theorems of probability theory*. Oxford Science Publications, Oxford, England, 1995.

[23] G. Pisier. Remarques sur un resultat non publi'e de B. Maurey. *Sem. d'Analyse Fonctionelle*, 1(12):1980–81, 1981.

[24] R. Rivest. Learning decision lists. *Machine Learning*, 2(3):229–246, 1987.

[25] R. Schapire. Theoretical views of boosting. In *Proc. 10th ALT*, pages 12–24, 1999.

[26] R. Servedio. On PAC learning using Winnow, Perceptron, and a Perceptron-like algorithm. In *Proceedings of the Twelfth Annual Conference on Computational Learning Theory*, pages 296–307, 1999.

[27] R. Servedio. Smooth boosting and learning with malicious noise. *Journal of Machine Learning Research*, 4:633–648, 2003. Preliminary version in *Proc. COLT'01*.

[28] R. Servedio. Every linear threshold function has a low-weight approximator. In *Proceedings of the 21st Conference on Computational Complexity (CCC)*, pages 18–30, 2006.

[29] L. Valiant. Projection learning. *Machine Learning*, 37(2):115–130, 1999.
